# Benchmarking Non-Parametric Statistical Tests

**Mikaela Keller**[*]
IDIAP Research Institute
1920 Martigny
Switzerland
mkeller@idiap.ch

**Samy Bengio**
IDIAP Research Institute
1920 Martigny
Switzerland
bengio@idiap.ch

**Siew Yeung Wong**
IDIAP Research Institute
1920 Martigny
Switzerland
sywong@idiap.ch

## Abstract

Although non-parametric tests have already been proposed for that purpose, statistical significance tests for non-standard measures (different from the classification error) are less often used in the literature. This paper is an attempt at empirically verifying how these tests compare with more classical tests, on various conditions. More precisely, using a very large dataset to estimate the whole "population", we analyzed the behavior of several statistical test, varying the class unbalance, the compared models, the performance measure, and the sample size. The main result is that providing big enough evaluation sets non-parametric tests are relatively reliable in all conditions.

## 1   Introduction

Statistical tests are often used in machine learning in order to assess the performance of a new learning algorithm or model over a set of benchmark datasets, with respect to the state-of-the-art solutions. Several researchers (see for instance [4] and [9]) have proposed statistical tests suited for 2-class classification tasks where the performance is measured in terms of the classification error (ratio of the number of errors and the number of examples), which enables the use of assumptions based on the fact that the error can be seen as a sum of random variables over the evaluation examples. On the other hand, various research domains prefer to measure the performance of their models using different indicators, such as the $F_1$ measure, used in information retrieval [11], described in Section 2.1. Most classical statistical tests cannot cope directly with such measure as the usual necessary assumptions are no longer correct, and non-parametric bootstrap-based methods are then used [5].

Since several papers already use these non-parametric tests [2, 1], we were interested in verifying empirically how reliable they were. For this purpose, we used a very large text categorization database (the extended Reuters dataset [10]), composed of more than 800000 examples, and concerning more than 100 categories (each document was labelled with one or more of these categories). We purposely set aside the largest part of the dataset and considered it as the whole population, while a much smaller part of it was used as a training set for the models. Using the large set aside dataset part, we *tested* the statistical test in the

---

[*]This work was supported in part by the Swiss NSF through the NCCR on IM2 and in part by the European PASCAL Network of Excellence, IST-2002-506778, through the Swiss OFES.

same spirit as was done in [4], by sampling evaluation sets over which we observed the performance of the models and the behavior of the significance test.

Following the taxonomy of questions of interest defined by Dieterich in [4], we can differentiate between statistical tests that analyze learning algorithms and statistical tests that analyze classifiers. In the first case, one intends to be robust to possible variations of the train and evaluation sets, while in the latter, one intends to only be robust to variations of the evaluation set. While the methods discussed in this paper can be applied alternatively to both approaches, we concentrate here on the second one, as it is more tractable (for the empirical section) while still corresponding to real life situations where the training set is fixed and one wants to compare two solutions (such as during a competition).

In order to conduct a thorough analysis, we tried to vary the evaluation set size, the class unbalance, the error measure, the statistical test itself (with its associated assumptions), and even the *closeness* of the compared learning algorithms. This paper, and more precisely Section 3, is a detailed account of this analysis. As it will be seen empirically, the *closeness* of the compared learning algorithms seems to have an effect on the resulting quality of the statistical tests: comparing an MLP and an SVM yields less reliable statistical tests than comparing two SVMs with a different kernel. To the best of our knowledge, this has never been considered in the literature of statistical tests for machine learning.

## 2 A Statistical Significance Test for the Difference of $F_1$

Let us first remind the basic classification framework in which statistical significance tests are used in machine learning. We consider comparing two models $A$ and $B$ on a two-class classification task where the goal is to classify input examples $x_i$ into the corresponding class $y_i \in \{-1, 1\}$, using already trained models $f_A(x_i)$ or $f_B(x_i)$. One can estimate their respective performance on some test data by counting the number of utterances of each possible outcome: either the obtained class corresponds to the desired class, or not. Let $N_{e,A}$ (resp. $N_{e,B}$) be the number of errors of model $A$ (resp. $B$) and $N$ the total number of test examples; The difference between models $A$ and $B$ can then be written as

$$D = \frac{N_{e,A} - N_{e,B}}{N} \ . \tag{1}$$

The usual starting point of most statistical tests is to define the so-called *null hypothesis* $H_0$ which considers that the two models are equivalent, and then verifies how probable this hypothesis is. Hence, assuming that $D$ is an instance of some random variable $\mathbf{D}$ which follows some distribution, we are interested in

$$p\left(|\mathbf{D}| < |D|\right) < \alpha \tag{2}$$

where $\alpha$ represents the risk of selecting the *alternate hypothesis* (the two models are different) while the *null hypothesis* is in fact true. This can in general be estimated easily when the distribution of $\mathbf{D}$ is known. In the simplest case, known as the *proportion test*, one assumes (reasonably) that the decision taken by each model on each example can be modeled by a Bernoulli, and further assumes that the errors of the models are independent. This is in general wrong in machine learning since the evaluation sets are the same for both models. When $N$ is large, this leads to estimate $\mathbf{D}$ as a Normal distribution with zero mean and standard deviation $\sigma_D$

$$\sigma_D = \sqrt{\frac{2\bar{C}(1 - \bar{C})}{N}} \tag{3}$$

where $\bar{C} = \frac{N_{e,A} + N_{e,B}}{2N}$ is the average classification error. In order to get rid of the wrong independence assumption between the errors of the models, the McNemar test [6] concentrates on examples which were differently classified by the two compared models. Following the notation of [4], let $N_{01}$ be the number of examples misclassified by model $A$ but not

by model $B$ and $N_{10}$ the number of examples misclassified by model $B$ but not by model $A$. It can be shown that the following statistics is approximatively distributed as a $\chi^2$ with 1 degree of freedom:

$$z = \frac{(|N_{01} - N_{10}| - 1)^2}{N_{01} + N_{10}}. \tag{4}$$

More recently, several other statistical tests have been proposed, such as the 5x2cv method [4] or the variance estimate proposed in [9], which both claim to better estimate the distribution of the errors (and hence the confidence on the statistical significance of the results). Note however that these solutions assume that the error of one model is the average of some random variable (the error) estimated on each example. Intuitively, it will thus tend to be Normally distributed as $N$ grows, following the central limit theorem.

## 2.1 The $F_1$ Measure

Text categorization is the task of assigning one or several categories, among a predefined set of $K$ categories, to textual documents. As explained in [11], text categorization is usually solved as $K$ 2-class classification problems, in a one-against-the-others approach. In this field two measures are considered of importance:

$$\text{Precision} = \frac{N_{tp}}{N_{tp} + N_{fp}}, \quad \text{and} \quad \text{Recall} = \frac{N_{tp}}{N_{tp} + N_{fn}},$$

where for each category $N_{tp}$ is the number of true positives (documents belonging to the category that were classified as such), $N_{fp}$ the number of false positives (documents out of this category but classified as being part of it) and $N_{fn}$ the number of false negatives (documents from the category classified as out of it). Precision and Recall are effectiveness measures, *i.e.* inside $[0, 1]$ interval, the closer to 1 the better. For each category $k$, $\text{Precision}_k$ measures the proportion of documents of the class among the ones considered as such by the classifier and $\text{Recall}_k$ the proportion of documents of the class correctly classified.

To summarize these two values, it is common to consider the so-called $F_1$ measure [12], often used in domains such as information retrieval, text categorization, or vision processing. $F_1$ can be described as the inverse of the harmonic mean of Precision and Recall:

$$F_1 = \left( \frac{1}{2} \left[ \frac{1}{\text{Recall}} + \frac{1}{\text{Precision}} \right] \right)^{-1} = \frac{2 \cdot \text{Precision} \cdot \text{Recall}}{\text{Precision} + \text{Recall}} = \frac{2N_{tp}}{2N_{tp} + N_{fn} + N_{fp}}. \tag{5}$$

Let us consider two models $A$ and $B$, which achieve a performance measured by $F_{1,A}$ and $F_{1,B}$ respectively. The difference $dF_1 = F_{1,A} - F_{1,B}$ does not fit the assumptions of the tests presented earlier. Indeed, it cannot be decomposed into a sum over the documents of independent random variables, since the numerator and the denominator of $dF_1$ are non constant sums over documents of independent random variables. For the same reason $F_1$, while being a proportion, cannot be considered as a random variable following a Normal distribution for which we could easily estimate the variance.

An alternative solution to measure the statistical significance of $dF_1$ is based on the Bootstrap Percentile Test proposed in [5]. The idea of this test is to approximate the unknown distribution of $dF_1$ by an estimate based on bootstrap replicates of the data.

## 2.2 Bootstrap Percentile Test

Given an evaluation set of size $N$, one draws, *with replacement*, $N$ samples from it. This gives the first bootstrap replicate $B_1$, over which one can compute the statistics of interest,

$dF_{1,B_1}$. Similarly, one can create as many bootstrap replicates $B_n$ as needed, and for each, compute $dF_{1,B_n}$. The higher $n$ is, the more precise should be the statistical test. Literature [3] suggests to create at least $\frac{50}{\alpha}$ replicates where $\alpha$ is the level of the test; for the smallest $\alpha$ we considered (0.01), this amounts to 5000 replicates. These 5000 estimates $dF_{1,B_i}$ represent the non-parametric distribution of the random variable $\mathbf{dF_1}$. From it, one can for instance consider an interval $[a, b]$ such that $p(a < \mathbf{dF_1} < b) = 1 - \alpha$ centered around the mean of $p(\mathbf{dF_1})$. If 0 lies outside this interval, one can say that $dF_1 = 0$ is not among the most probable results, and thus reject the null hypothesis.

## 3  Analysis of Statistical Tests

We report in this section an analysis of the bootstrap percentile test, as well as other more classical statistical tests, based on a real large database. We first describe the database itself and the protocol we used for this analysis, and then provide results and comments.

### 3.1  Database, Models and Protocol

All the experiments detailed in this paper are based on the very large RCV1 Reuters dataset [10], which contains up to 806,791 documents. We divided it as follows: 798,809 documents were kept aside and any statistics computed over this set $D_{true}$ was considered as being the *truth* (*ie* a very good estimate of the actual value); the remaining 7982 documents were used as a training set $D_{tr}$ (to train models $A$ and $B$). There was a total of 101 categories and each document was labeled with one or more of these categories.

We first extracted the dictionary from the training set, removed stop-words and applied stemming to it, as normally done in text categorization. Each document was then represented as a bag-of-words using the usual $tfidf$ coding. We trained three different models: a linear Support Vector Machine (SVM), a Gaussian kernel SVM, and a multi-layer perceptron (MLP). There was one model for each category for the SVMs, and a single MLP for the 101 categories. All models were properly tuned using cross-validation on the training set.

Using the notation introduced earlier, we define the following competing hypotheses:
$H_0 : |dF_1| = 0$ and $H_1 : |dF_1| > 0$. We further define the level of the test $\alpha = p(\text{Reject } H_0 | H_0)$, where $\alpha$ takes on values 0.01, 0.05 and 0.1. Table 1 summarizes the possible outcomes of a statistical test. With that respect, rejecting $H_0$ means that one is confident with $(1 - \alpha) \cdot 100\%$ that $H_0$ is really false.

Table 1: Various outcomes of a statistical test, with $\alpha = p(\text{Type I error})$.

| Truth | Decision | |
|---|---|---|
| | Reject $H_0$ | Accept $H_0$ |
| $H_0$ | Type I error | OK |
| $H_1$ | OK | Type II error |

In order to assess the performance of the statistical tests on their Type I error, also called Size of the test, and on their Power $= 1-$ Type II error, we used the following protocol.

For each category $C_i$, we sampled over $D_{true}$, $S$ (500) evaluation sets $D_{te}^s$ of $N$ documents, ran the significance test over each $D_{te}^s$ and computed the proportion of sets for which $H_0$ was rejected given that $H_0$ was true over $D_{true}$ (*resp.* $H_0$ was false over $D_{true}$), which we note $\alpha_{true}$ (*resp.* $\pi$).

We used $\alpha_{true}$ as an estimate of the significance test's probability of making a Type I error

and $\pi$ as an estimate of the significance test's Power. When $\alpha_{true}$ is higher than the $\alpha$ fixed by the statistical test, the test underestimates Type I error, which means we should not rely on its decision regarding the superiority of one model over the other. Thus, we consider that the significance test fails. On the contrary, $\alpha_{true} < \alpha$ yields a pessimistic statistical test that decides correctly $H_0$ more often than predicted.

Furthermore we would like to favor significance tests with a high $\pi$, since the Power of the test reflects its ability to reject $H_0$ when $H_0$ is false.

## 3.2 Summary of Conditions

In order to verify the sensitivity of the analyzed statistical tests to several conditions, we varied the following parameters:

- the value of $\alpha$: it took on values in $\{0.1, 0.05, 0.01\}$;
- the two compared models: there were three models, two of them were of the same family (SVMs), hence optimizing the same criterion, while the third one was an MLP. Most of the times the two SVMs gave very similar results, (probably because the optimal capacity for this problem was near linear), while the MLP gave poorer results on average. The point here was to verify whether the test was sensitive to the *closeness* of the tested models (although a more formal definition of *closeness* should certainly be devised);
- the evaluation sample size: we varied it from small sizes (100) up to larger sizes (6000) to see the robustness of the statistical test to it;
- the class unbalance: out of the 101 categories of the problem, most of them resulted in highly unbalanced tasks, often with a ratio of 10 to 100 between the two classes. In order to experiment with more balanced tasks, we artificially created *meta-categories*, which were random aggregations of normal categories that tended to be more balanced;
- the tested measure: our initial interest was to directly test $dF_1$, the difference of $F_1$, but given poor initial results, we also decided to assess $dCerr$, the difference of classification errors, in order to see whether the tests were sensitive to the measure itself;
- the statistical test: on top of the bootstrap percentile test, we also analyzed the more classical *proportion test* and *McNemar test*, both of them only on $dCerr$ (since they were not adapted to $dF_1$).

## 3.3 Results

Figure 1 summarizes the results for the Size of the test estimates. All graphs show $\alpha_{true}$, the number of times the test rejected $H_0$ while $H_0$ was true, for a fixed $\alpha = 0.05$, with respect to the sample size, for various statistical tests and tested measures.

Figure 2 shows the obtained results for the Power of the test estimates. The proportion of evaluation sets over which the significance test (with $\alpha = 0.05$) rejected $H_0$ when indeed $H_0$ was false, is plotted against the evaluation set size.

Figures 1(a) and 2(a) show the results for balanced data (where the positive and negative examples were approximatively equally present in the evaluation set) when comparing two different models (an SVM and an MLP).

Figures 1(b) and 2(b) show the results for unbalanced data when comparing two different models.

Figures 1(c) and 2(c) show the results for balanced data when comparing two similar models (a linear SVM and a Gaussian SVM) for balanced data, and finally Figures 1(d) and 2(d)

show the results for unbalanced data and two similar models.

Note that each point in the graphs was computed over a different number of samples, since *eg* over the (500 evaluation sets $\times$ 101 categories) experiments only those for which $H_0$ was true in $D_{true}$ were taken into account in the computation of $\alpha_{true}$.

When the proportion of $H_0$ true in $D_{true}$ equals 0 (*resp.* the proportion of $H_0$ false in $D_{true}$ equals 0), $\alpha_{true}$ (*resp.* $\pi$) is set to -1. Hence, for instance the first points ($\{100, \ldots, 1000\}$) of Figures 2(c) and 2(d) were computed over only 500 evaluation sets on which respectively the same categorization task was performed. This makes these points unreliable. See [8] for more details.

For each of the Size's graphs, when the curves are over the 0.05 line, we can state that the statistical test is optimistic, while when it is below the line, the statistical test is pessimistic. As already explained, a pessimistic test should be favored whenever possible.

Several interesting conclusions can be drawn from the analysis of these graphs. First of all, as expected, most of the statistical tests are positively influenced by the size of the evaluation set, in the sense that their $\alpha_{true}$ value converges to $\alpha$ for large sample sizes [1].

On the available results, the McNemar test and the bootstrap test over $dCerr$ have a similar performance. They are always pessimistic even for small evaluation set sizes, and tend to the expected $\alpha$ values when the models compared on balanced tasks are dissimilar. They have also a similar performance in Power over all the different conditions, higher in general when comparing very different models.

When the compared models are similar, the bootstrap test over $dF_1$ has a pessimistic behavior even on quite small evaluation sets. However, when the models are really different the bootstrap test over $dF_1$ is on average always optimistic. Note nevertheless that most of the points in Figures 1(a) and 1(b) have a standard deviation $std$, over the categories, such that $\alpha_{true} - std < \alpha$ (see [8] for more details). Another interesting point is that in the available results for the Power, the $dF_1$'s bootstrap test have relatively high values with respect to the other tests.

The proportion test have in general, on the available results, a more conservative behavior than the McNemar test and the $dCerr$ bootstrap test. It has more pessimistic results and less Power. It is too often prone to "Accept $H_0$", *ie* to conclude that the compared models have an equivalent performance, whether it is true or not. This results seem to be consistent with those of [4] and [9]. However, when comparing *close* models in a small unbalanced evaluation set (Figure 1(d)), this conservative behavior is not present.

To summarize the findings, the bootstrap-based statistical test over $dCerr$ obtained a good performance in Size comparable to the one of the McNemar test in all conditions. However both significance test performances in Power are low even for big evaluation sets in particular when the compared models are close. The bootstrap-based statistical test over $dF_1$ has higher Power than the other compared tests, however it must be emphasized that it is slightly over-optimistic in particular for small evaluation sets. Finally, when applying the proportion test over unbalanced data for *close* models we obtained an optimistic behavior, untypical of this usually conservative test.

## 4 Conclusion

In this paper, we have analyzed several parametric and non-parametric statistical tests for various conditions often present in machine learning tasks, including the class balancing, the performance measure, the size of the test sets, and the *closeness* of the compared mod-

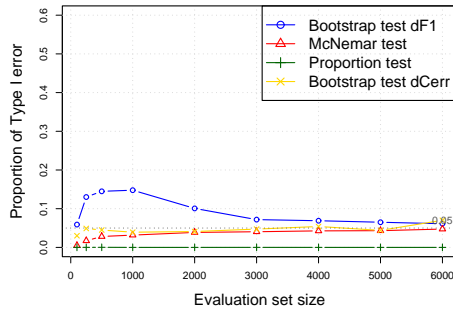

(a) Linear SVM vs MLP - Balanced data

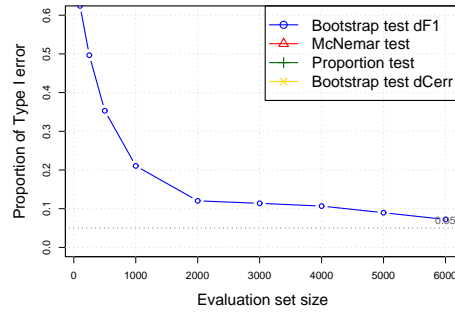

(b) Linear SVM vs MLP - Unbalanced data

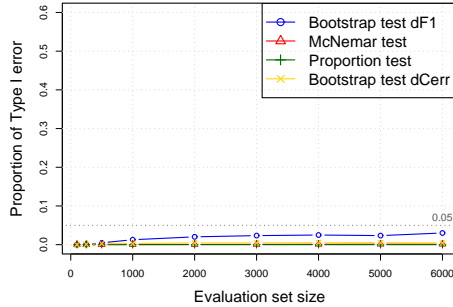

(c) Linear vs RBF SVMs - Balanced data

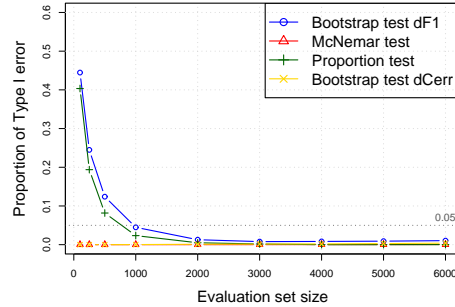

(d) Linear vs RBF SVMs - Unbalanced data

Figure 1: Several statistical tests comparing Linear SVM vs MLP or vs RBF SVM. The proportion of Type I error equals -1, in Figure 1(b), when there was no data to compute the proportion (*ie* $H_0$ was always false).

els. More particularly, we were concerned by the quality of non-parametric tests since in some cases (when using more complex performance measures such as $F_1$), they are the only available statistical tests.

Fortunately, most statistical tests performed reasonably well (in the sense that they were more often pessimistic than optimistic in their decisions) and larger test sets always improved their performance. Note however that for $dF_1$ the only available statistical test was too optimistic although consistant for different levels. An unexpected result was that the rather conservative proportion test used over unbalanced data for *close* models yielded an optimistic behavior.

It has to be noted that recently, a probabilistic interpretation of $F_1$ was suggested in [7], and a comparison with bootstrap-based tests should be worthwhile.

## Footnotes

[1]Note that the same is true for the variance of $\alpha_{true}(\rightarrow 0)$, and this for any of the $\alpha$ values tested.

# References

[1] M. Bisani and H. Ney. Bootstrap estimates for confidence intervals in ASR performance evaluation. In *Proceedings of ICASSP*, 2004.

[2] R. M. Bolle, N. K. Ratha, and S. Pankanti. Error analysis of pattern recognition systems - the subsets bootstrap. *Computer Vision and Image Understanding*, 93:1–33, 2004.

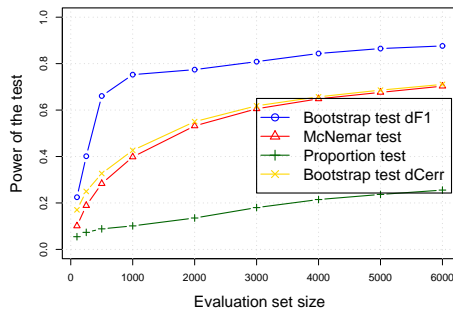

(a) Linear SVM vs MLP - Balanced data

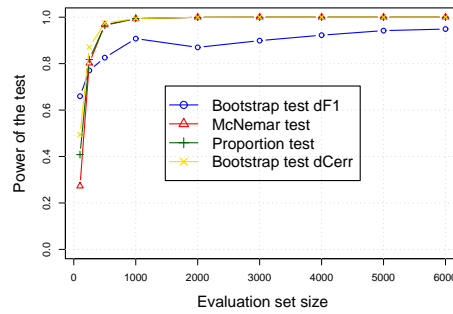

(b) Linear SVM vs MLP - Unbalanced data

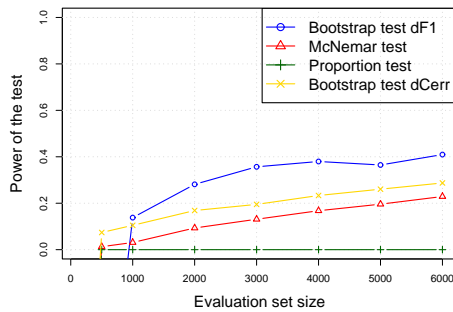

(c) Linear vs RBF SVMs - Balanced data

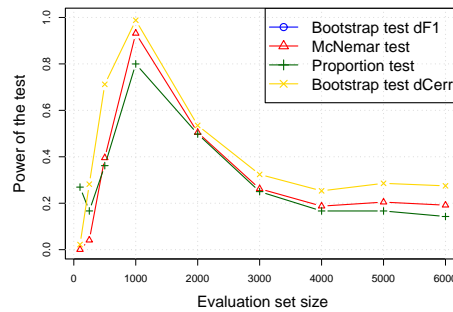

(d) Linear vs RBF SVMs - Unbalanced data

Figure 2: Power of several statistical tests comparing Linear SVM vs MLP or vs RBF SVM. The power equals -1, in Figures 2(c) and 2(d), when there was not data to compute the proportion (*ie* $H_1$ was never true).

[3] A. C. Davison and D. V. Hinkley. *Bootstrap methods and their application*. Cambridge University Press, 1997.

[4] T.G. Dietterich. Approximate statistical tests for comparing supervised classification learning algorithms. *Neural Computation*, 10(7):1895–1924, 1998.

[5] B. Efron and R. Tibshirani. *An Introduction to the Bootstrap*. Chapman and Hall, 1993.

[6] B. S. Everitt. *The analysis of contingency tables*. Chapman and Hall, 1977.

[7] C. Goutte and E. Gaussier. A probabilistic interpretation of precision, recall and F-score, with implication for evaluation. In *Proceedings of ECIR*, pages 345–359, 2005.

[8] M. Keller, S. Bengio, and S. Y. Wong. Surprising Outcome While Benchmarking Statistical Tests. IDIAP-RR 38, IDIAP, 2005.

[9] Claude Nadeau and Yoshua Bengio. Inference for the generalization error. *Machine Learning*, 52(3):239–281, 2003.

[10] T.G. Rose, M. Stevenson, and M. Whitehead. The Reuters Corpus Volume 1 - from yesterday's news to tomorrow's language resources. In *Proceedings of the 3rd Int. Conf. on Language Resources and Evaluation*, 2002.

[11] F. Sebastiani. Machine learning in automated text categorization. *ACM Computing Surveys*, 34(1):1–47, 2002.

[12] C. J. van Rijsbergen. *Information Retrieval*. Butterworths, London, UK, 1975.
